# Multi-label Multiple Kernel Learning

**Shuiwang Ji**
Arizona State University
Tempe, AZ 85287
shuiwang.ji@asu.edu

**Liang Sun**
Arizona State University
Tempe, AZ 85287
sun.liang@asu.edu

**Rong Jin**
Michigan State University
East Lansing, MI 48824
rongjin@cse.msu.edu

**Jieping Ye**
Arizona State University
Tempe, AZ 85287
jieping.ye@asu.edu

## Abstract

We present a multi-label multiple kernel learning (MKL) formulation in which the data are embedded into a low-dimensional space directed by the instance-label correlations encoded into a hypergraph. We formulate the problem in the kernel-induced feature space and propose to learn the kernel matrix as a linear combination of a given collection of kernel matrices in the MKL framework. The proposed learning formulation leads to a non-smooth min-max problem, which can be cast into a semi-infinite linear program (SILP). We further propose an approximate formulation with a guaranteed error bound which involves an unconstrained convex optimization problem. In addition, we show that the objective function of the approximate formulation is differentiable with Lipschitz continuous gradient, and hence existing methods can be employed to compute the optimal solution efficiently. We apply the proposed formulation to the automated annotation of *Drosophila* gene expression pattern images, and promising results have been reported in comparison with representative algorithms.

## 1   Introduction

Spectral graph-theoretic methods have been used widely in unsupervised and semi-supervised learning recently. In this paradigm, a weighted graph is constructed for the data set, where the nodes represent the data points and the edge weights characterize the relationships between vertices. The structural and spectral properties of graph can then be exploited to perform the learning task. One fundamental limitation of using traditional graphs for this task is that they can only represent pairwise relationships between data points, and hence higher-order information cannot be captured [1]. Hypergraphs [1, 2] generalize traditional graphs by allowing edges, called hyperedges, to connect more than two vertices, thereby being able to capture the relationships among multiple vertices.

In this paper, we propose to use a hypergraph to capture the correlation information for multi-label learning [3]. In particular, we propose to construct a hypergraph for multi-label data in which all data points annotated with a common label are included in a hyperedge, thereby capturing the similarity among data points with a common label. By exploiting the spectral properties of the constructed hypergraph, we propose to embed the multi-label data into a lower-dimensional space in which data points with a common label tend to be close to each other. We formulate the multi-label learning problem in the kernel-induced feature space, and show that the well-known kernel canonical correlation analysis (KCCA) [4] is a special case of the proposed framework. As the kernel plays an essential role in the formulation, we propose to learn the kernel matrix as a linear combination of a given collection of kernel matrices in the multiple kernel learning (MKL) framework. The resulting

formulation involves a non-smooth min-max problem, and we show that it can be cast into a semi-infinite linear program (SILP). To further improve the efficiency and reduce the non-smoothness effect of the SILP formulation, we propose an approximate formulation by introducing a smoothing term into the original problem. The resulting formulation is unconstrained and convex. In addition, the objective function of the approximate formulation is shown to be differentiable with Lipschitz continuous gradient. We can thus employ the Nesterov's method [5, 6], which solves smooth convex problems with the optimal convergence rate, to compute the solution efficiently.

We apply the proposed formulation to the automated annotation of *Drosophila* gene expression pattern images, which document the spatial and temporal dynamics of gene expression during *Drosophila* embryogenesis [7]. Comparative analysis of such images can potentially reveal new genetic interactions and yield insights into the complex regulatory networks governing embryonic development. To facilitate pattern comparison and searching, groups of images are annotated with a variable number of labels by human curators in the Berkeley *Drosophila* Genome Project (BDGP) high-throughput study [7]. However, the number of available images produced by high-throughput *in situ* hybridization is now rapidly increasing. It is therefore tempting to design computational methods to automate this task [8]. Since the labels are associated with groups of a variable number of images, we propose to extract invariant features from each image and construct kernels between groups of images by employing the vocabulary-guided pyramid match algorithm [9]. By applying various local descriptors, we obtain multiple kernel matrices and the proposed multi-label MKL formulation is applied to obtain an optimal kernel matrix for the low-dimensional embedding. Experimental results demonstrate the effectiveness of the kernel matrices obtained by the proposed formulation. Moreover, the approximate formulation is shown to yield similar results to the original formulation, while it is much more efficient.

## 2 Multi-label Learning with Hypergraph

An essential issue in learning from multi-label data is how to exploit the correlation information among labels. We propose to capture such information through a hypergraph as described below.

### 2.1 Hypergraph Spectral Learning

Hypergraphs generalize traditional graphs by allowing hyperedges to connect more than two vertices, thus capturing the joint relationships among multiple vertices. We propose to construct a hypergraph for multi-label data in which each data point is represented as a vertex. To document the joint similarity among data points annotated with a common label, we propose to construct a hyperedge for each label and include all data points annotated with a common label into one hyperedge. Following the spectral graph embedding theory [10], we propose to compute the low-dimensional embedding through a linear transformation $W$ by solving the following optimization problem:

$$\min_{W} \quad \text{tr}\left(W^T \phi(X) \mathcal{L} \phi(X)^T W\right) \tag{1}$$

$$\text{subject to} \quad W^T \left(\phi(X)\phi(X)^T + \lambda I\right) W = I,$$

where $\phi(X) = [\phi(x_1), \cdots, \phi(x_n)]$ is the data matrix consisting of $n$ data points in the feature space, $\phi$ is the feature mapping, $\mathcal{L}$ is the normalized Laplacian matrix derived from the hypergraph, and $\lambda > 0$ is the regularization parameter. In this formulation, the instance-label correlations are encoded into $\mathcal{L}$ through the hypergraph, and data points sharing a common label tend to be close to each other in the embedded space.

It follows from the *representer theorem* [11] that $W = \phi(X)B$ for some matrix $B \in \mathbb{R}^{n \times k}$ where $k$ is the number of labels. By noting that $\mathcal{L} = I - C$ for some matrix $C$, the problem in Eq. (1) can be reformulated as

$$\max_{B} \quad \text{tr}\left(B^T(KCK)B\right) \tag{2}$$

$$\text{subject to} \quad B^T(K^2 + \lambda K)B = I,$$

where $K = \phi(X)^T \phi(X)$ is the kernel matrix. Kernel canonical correlation analysis (KCCA) [4] is a widely-used method for dimensionality reduction. It can be shown [4] that KCCA is obtained by substituting $C = Y^T(YY^T)^{-1}Y$ in Eq. (2) where $Y \in \mathbb{R}^{k \times n}$ is the label indicator matrix. Thus, KCCA is a special case of the proposed formulation.

## 2.2 A Semi-infinite Linear Program Formulation

It follows from the theory of kernel methods [11] that the kernel $K$ in Eq. (2) uniquely determines the feature mapping $\phi$. Thus, kernel selection (learning) is one of the central issues in kernel methods. Following the MKL framework [12], we propose to learn an optimal kernel matrix by integrating multiple candidate kernel matrices, that is,

$$K \in \mathcal{K} = \left\{ K = \sum_{j=1}^{p} \theta_j K_j \,\middle|\, \theta^T e = 1, \ \theta \geq 0 \right\}, \tag{3}$$

where $\{K_j\}_{j=1}^{p}$ are the $p$ candidate kernel matrices, $\{\theta_j\}_{j=1}^{p}$ are the weights for the linear combination, and $e$ is the vector of all ones of length $p$. We have assumed in Eq. (3) that all the candidate kernel matrices are normalized to have a unit trace value. It has been shown [8] that the optimal weights maximizing the objective function in Eq. (2) can be obtained by solving a semi-infinite linear program (SILP) [13] in which a linear objective is optimized subject to an infinite number of linear constraints, as summarized in the following theorem:

**Theorem 2.1.** *Given a set of $p$ kernel matrices $\{K_j\}_{j=1}^{p}$, the optimal kernel matrix in $\mathcal{K}$ that maximizes the objective function in Eq. (2) can be obtained by solving the following SILP problem:*

$$\max_{\theta, \gamma} \quad \gamma \tag{4}$$

$$subject \ to \quad \theta \geq 0, \ \theta^T e = 1, \ \sum_{j=1}^{p} \theta_j S_j(Z) \geq \gamma, \ for \ all \ Z \in \mathbb{R}^{n \times k}, \tag{5}$$

*where $S_j(Z)$, for $j = 1, \cdots, p$, is defined as*

$$S_j(Z) = \sum_{i=1}^{k} \left( \frac{1}{4} z_i^T z_i + \frac{1}{4\lambda} z_i^T K_j z_i - z_i^T h_i \right), \tag{6}$$

$Z = [z_1, \cdots, z_k]$, $H$ is obtained from $C$ such that $HH^T = C$, and $H = [h_1, \cdots, h_k]$.

Note that the matrix $C$ is symmetric and positive semidefinite. Moreover, for the $\mathcal{L}$ considered in this paper, we have $\text{rank}(C) = k$. Hence, $H \in \mathbb{R}^{n \times k}$ is always well-defined. The SILP formulation in Theorem 2.1 can be solved by the column generation technique as in [14].

## 3   The Approximate Formulation

The multi-label kernel learning formulation proposed in Theorem 2.1 involves optimizing a linear objective subject to an infinite number of constraints. The column generation technique used to solve this problem adds constraints to the problem successively until all the constraints are satisfied. Since the convergence rate of this algorithm is slow, the problem solved at each iteration may involve a large number of constraints, and hence is computationally expensive. In this section, we propose an approximate formulation by introducing a smoothing term into the original problem. This results in an unconstrained and smooth convex problem. We propose to employ existing methods to solve the smooth convex optimization problem efficiently in the next section.

By rewriting the formulation in Theorem 2.1 as

$$\max_{\theta:\theta^T e=1, \theta \geq 0} \min_{Z} \sum_{j=1}^{p} \theta_j S_j(Z)$$

and exchanging the minimization and maximization, the SILP formulation can be expressed as

$$\min_{Z} f(Z) \tag{7}$$

where $f(Z)$ is defined as

$$f(Z) = \max_{\theta:\theta^T e=1, \theta \geq 0} \sum_{j=1}^{p} \theta_j S_j(Z). \tag{8}$$

The maximization problem in Eq. (8) with respect to $\theta$ leads to a non-smooth objective function for $f(Z)$. To reduce this effect, we introduce a smoothing term and modify the objective to $f_\mu(Z)$ as

$$f_\mu(Z) = \max_{\theta:\theta^T e=1, \theta \geq 0} \left\{ \sum_{j=1}^{p} \theta_j S_j(Z) - \mu \sum_{j=1}^{p} \theta_j \log \theta_j \right\}, \tag{9}$$

where $\mu$ is a positive constant controlling the approximation. The following lemma shows that the problem in Eq. (9) can be solved analytically:

**Lemma 3.1.** *The optimization problem in Eq. (9) can be solved analytically, and the optimal value can be expressed as*

$$f_\mu(Z) = \mu \log \left( \sum_{j=1}^{p} \exp \left( \frac{1}{\mu} S_j(Z) \right) \right). \tag{10}$$

*Proof.* Define the Lagrangian function for the optimization problem in Eq. (9) as

$$L = \sum_{j=1}^{p} \theta_j S_j(Z) - \mu \sum_{j=1}^{p} \theta_j \log \theta_j + \sum_{j=1}^{p} \alpha_j \theta_j + \left( \sum_{j=1}^{p} \theta_j - 1 \right) \beta, \tag{11}$$

where $\{\alpha_j\}_{j=1}^{p}$ and $\beta$ are Lagrangian dual variables. Taking the derivative of the Lagrangian function with respect to $\theta_j$ and setting it to zero, we obtain that $\theta_j = \exp \left( \frac{1}{\mu} (S_j(Z) + \alpha_j + \beta - \mu) \right)$. It follows from the complementarity condition that $\alpha_j \theta_j = 0$ for $j = 1, \cdots, p$. Since $\theta_j \neq 0$, we have $\alpha_j = 0$ for $j = 1, \cdots, p$. By removing $\{\alpha_j\}_{j=1}^{p}$ and substituting $\theta_j$ into the objective function in Eq. (9), we obtain that $f_\mu(Z) = \mu - \beta$. Since $\mu - \beta = S_j(Z) - \mu \log \theta_j$, we have

$$\theta_j = \exp \left( (S_j(Z) - f_\mu(Z))/\mu \right). \tag{12}$$

Following $1 = \sum_{j=1}^{p} \theta_j = \sum_{j=1}^{p} \exp \left( (S_j(Z) - f_\mu(Z))/\mu \right)$, we obtain Eq. (10). □

The above discussion shows that we can approximate the original non-smooth constrained min-max problem in Eq. (7) by the following smooth unconstrained minimization problem:

$$\min_Z f_\mu(Z), \tag{13}$$

where $f_\mu(Z)$ is defined in Eq. (10). We show in the following two lemmas that the approximate formulation in Eq. (13) is convex and has a guaranteed approximation bound controlled by $\mu$.

**Lemma 3.2.** *The problem in Eq. (13) is a convex optimization problem.*

*Proof.* The optimization problem in Eq. (13) can be expressed equivalently as

$$\min_{Z, \{u_j\}_{j=1}^{p}, \{v_j\}_{j=1}^{p}} \mu \log \left( \sum_{j=1}^{p} \exp \left( u_j + v_j - \sum_{i=1}^{k} z_i^T h_i \right) \right) \tag{14}$$

$$\text{subject to} \quad \mu u_j \geq \frac{1}{4} \sum_{i=1}^{k} z_i^T z_i, \quad \mu v_j \geq \frac{1}{4\lambda} \sum_{i=1}^{k} z_i^T K_j z_i, \ \ j = 1, \cdots, p.$$

Since the log-exponential-sum function is a convex function and the two constraints are second-order cone constraints, the problem in Eq. (13) is a convex optimization problem. □

**Lemma 3.3.** *Let $f(Z)$ and $f_\mu(Z)$ be defined as above. Then we have $f_\mu(Z) \geq f(Z)$ and $|f_\mu(Z) - f(Z)| \leq \mu \log p$.*

*Proof.* The term $-\sum_{j=1}^{p} \theta_j \log \theta_j$ defines the entropy of $\{\theta_j\}_{j=1}^{p}$ when it is considered as a probability distribution, since $\theta \geq 0$ and $\theta^T e = 1$. Hence, this term is non-negative and $f_\mu(Z) \geq f(Z)$. It is known from the property of entropy that $-\sum_{j=1}^{p} \theta_j \log \theta_j$ is maximized with a uniform $\{\theta_j\}_{j=1}^{p}$, i.e., $\theta_j = \frac{1}{p}$ for $j = 1, \cdots, p$. Thus, we have $-\sum_{j=1}^{p} \theta_j \log \theta_j \leq \log p$ and $|f_\mu(Z) - f(Z)| = -\mu \sum_{j=1}^{p} \theta_j \log \theta_j \leq \mu \log p$. This completes the proof of the lemma. □

# 4 Solving the Approximate Formulation Using the Nesterov's Method

The Nesterov's method (known as "the optimal method" in [5]) is an algorithm for solving smooth convex problems with the optimal rate of convergence. In this method, the objective function needs to be differentiable with Lipschitz continuous gradient. In order to apply this method to solve the proposed approximate formulation, we first compute the Lipschitz constant for the gradient of function $f_\mu(Z)$, as summarized in the following lemma:

**Lemma 4.1.** *Let $f_\mu(Z)$ be defined as in Eq. (10). Then the Lipschitz constant $L$ of the gradient of $f_\mu(Z)$ can be bounded from above as*

$$L \leq L_\mu, \tag{15}$$

*where $L_\mu$ is defined as*

$$L_\mu = \frac{1}{2} + \frac{1}{2\lambda} \max_{1 \leq j \leq p} \lambda_{max}(K_j) + \frac{1}{8\mu\lambda^2} tr(Z^T Z) \max_{1 \leq i,j \leq p} \lambda_{max}((K_i - K_j)(K_i - K_j)^T), \tag{16}$$

*and $\lambda_{max}(\cdot)$ denotes the maximum eigenvalue. Moreover, the distance from the origin to the optimal set of $Z$ can be bounded as $tr(Z^T Z) \leq R_\mu^2$ where $R_\mu^2$ is defined as*

$$R_\mu^2 = \sum_{i=1}^{k} \left( ||[C_j]_i||_2 + \sqrt{4\mu \log p + tr\left( C_j^T \left[ I + \frac{1}{\lambda}K_j \right] C_j \right)} \right)^2, \tag{17}$$

$C_j = 2 \left( I + \frac{1}{\lambda}K_j \right)^{-1} H$ *and $[C_j]_i$ denotes the $i$th column of $C_j$.*

*Proof.* To compute the Lipschitz constant for the gradient of $f_\mu(Z)$, we first compute the first and second order derivatives as follows:

$$\nabla f_\mu(Z) = \sum_{j=1}^{p} g_j \left( \frac{\text{vec}(Z)}{2} + \frac{\text{vec}(K_j Z)}{2\lambda} \right) - \text{vec}(H), \tag{18}$$

$$\nabla^2 f_\mu(Z) = \frac{1}{2}I + \sum_{j=1}^{p} \frac{g_j}{2\lambda} D_k(K_j)$$

$$+ \frac{1}{8\mu} \sum_{i,j=1}^{p} g_i g_j \left( \frac{\text{vec}(K_i Z)}{\lambda} - \frac{\text{vec}(K_j Z)}{\lambda} \right) \left( \frac{\text{vec}(K_i Z)}{\lambda} - \frac{\text{vec}(K_j Z)}{\lambda} \right)^T, \tag{19}$$

where $\text{vec}(\cdot)$ converts a matrix into a vector, $D_k(K_j) \in \mathbb{R}^{(n \times k) \times (n \times k)}$ is a block diagonal matrix with the $k$th diagonal block as $K_j$, and $g_j = \exp(S_j(Z)/\mu)/\sum_{i=1}^{p} \exp(S_i(Z)/\mu)$. Then we have

$$L \leq \frac{1}{2} + \frac{1}{2\lambda} \max_{1 \leq j \leq p} \lambda_{max}(K_j) + \frac{1}{8\mu\lambda^2} \max_{1 \leq i,j \leq p} tr(Z^T (K_i - K_j)(K_i - K_j)^T Z) \leq L_\mu.$$

where $L_\mu$ is defined in Eq. (16).

We next derive the upper bound for $tr(Z^T Z)$. To this end, we first rewrite $S_j(Z)$ as

$$S_j(Z) = \frac{1}{4}tr\left( (Z - C_j)^T \left[ I + \frac{1}{\lambda}K_j \right] (Z - C_j) \right) - \frac{1}{4}tr\left( C_j^T \left[ I + \frac{1}{\lambda}K_j \right] C_j \right).$$

Since $\min f_\mu(Z) \leq f_\mu(0) = \mu \log p$, and $f_\mu(Z) \geq S_j(Z)$, we have $S_j(Z) \leq \mu \log p$ for $j = 1, \cdots, p$. It follows that $\frac{1}{4}tr\left( (Z - C_j)^T (Z - C_j) \right) \leq \mu \log p + \frac{1}{4}tr\left( C_j^T \left[ I + \frac{1}{\lambda}K_j \right] C_j \right)$. By using this inequality, it can be verified that $tr(Z^T Z) \leq R_\mu^2$ where $R_\mu^2$ is defined in Eq. (17). $\square$

The Nesterov's method for solving the proposed approximate formulation is presented in Table 1. After the optimal $Z$ is obtained from the Nesterov's method, the optimal $\{\theta_j\}_{j=1}^{p}$ can be computed from Eq. (12). It follows from the convergence proof in [5] that after $N$ iterations, as long as $f_\mu(X^i) \leq f_\mu(X^0)$ for $i = 1, \cdots, N$, we have

$$f_\mu(Z^{N+1}) - f_\mu(Z^*) \leq \frac{4L_\mu R_\mu^2}{(N+1)^2}, \tag{20}$$

Table 1: The Nesterov's method for solving the proposed multi-label MKL formulation.

- Initialize $X^0 = Z^1 = Q^0 = \mathbf{0} \in \mathbb{R}^{n \times k}$, $t_0 = 1$, $L_0 = \frac{1}{2} + \frac{1}{2\lambda} \max_{1 \le j \le p} \lambda_{\max}(K_j)$, and $\mu = \frac{1}{N}$ where $N$ is the predefined number of iterations
- **for** $i = 1, \cdots, N$ **do**
    - Set $X^i = Z^i - \frac{1}{t_{i-1}}(Z^i + Q^{i-1})$
    - Compute $f_\mu(X^i)$ and $\nabla f_\mu(X^i)$
    - Set $L = L_{i-1}$
    - **while** $f_\mu(X^i - \nabla f_\mu(X^i)/L) > f_\mu(X^i) - \frac{1}{2L}\text{tr}((\nabla f_\mu(X^i))^T \nabla f_\mu(X^i))$ **do**
        - $L = L \times 2$
    - **end while**
    - Set $L_i = L$
    - Set $Z^{i+1} = X^i - \frac{1}{L_i}\nabla f_\mu(X^i), \quad Q^i = Q^{i-1} + \frac{t_{i-1}}{L_i}\nabla f_\mu(X^i)$
    - Set $t_i = \frac{1}{2}\left(1 + \sqrt{1 + 4t_{i-1}^2}\right)$
- **end for**

where $Z^* = \arg\min_Z f_\mu(Z)$. Furthermore, since $f_\mu(Z^{N+1}) \ge f(Z^{N+1})$ and $f_\mu(Z^*) \le f(Z^*) + \mu \log p$, we have

$$f(Z^{N+1}) - f(Z^*) \le \mu \log p + \frac{4L_\mu R_\mu^2}{(N+1)^2}. \tag{21}$$

By setting $\mu = O(1/N)$, we have that $L_\mu \propto O(1/\mu) \propto O(N)$. Hence, the convergence rate of the Nesterov's method is on the order of $O(1/N)$. This is significantly better than the convergence rates of $O(1/N^{1/3})$ and $O(1/N^{1/2})$ for the SILP and the gradient descent method, respectively.

## 5   Experiments

In this section, we evaluate the proposed formulation on the automated annotation of gene expression pattern images. The performance of the approximate formulation is also validated.

**Experimental Setup** The experiments use a collection of gene expression pattern images retrieved from the FlyExpress database (http://www.flyexpress.net). We apply nine local descriptors (SIFT, shape context, PCA-SIFT, spin image, steerable filters, differential invariants, complex filters, moment invariants, and cross correlation) on regular grids of 16 and 32 pixels in radius and spacing on each image. These local descriptors are commonly used in computer vision problems [15]. We also apply Gabor filters with different wavelet scales and filter orientations on each image to obtain global features of 384 and 2592 dimensions. Moreover, we sample the pixel values of each image to obtain features of 10240, 2560, and 640 dimensions. After generating the features, we apply the vocabulary-guided pyramid match algorithm [9] to construct kernels between the image sets. A total of 23 kernel matrices (2 grid size $\times$ 9 local descriptors + 2 Gabor + 3 pixel) are constructed. Then the proposed MKL formulation is employed to obtain the optimal integrated kernel matrix based on which the low-dimensional embedding is computed. We use the expansion-based approach (*star* and *clique*) to construct the hypergraph Laplacian, since it has been shown [1] that the Laplacians constructed in this way are similar to those obtained directly from a hypergraph. The performance of kernel matrices (either single or integrated) is evaluated by applying the support vector machine (SVM) for each term using the one-against-rest scheme. The F1 score is used as the performance measure, and both *macro*-averaged and *micro*-averaged F1 scores across labels are reported. In each case, the entire data set is randomly partitioned into training and test sets with a ratio of 1:1. This process is repeated ten times, and the averaged performance is reported.

**Performance Evaluation** It can be observed from Tables 2 and 3 that in terms of both macro and micro F1 scores, the kernels integrated by either star or clique expansions achieve the highest performance on almost all of the data sets. In particular, the integrated kernels outperform the best individual kernel significantly on all data sets. This shows that the proposed formulation is effective

Table 2: Performance of integrated kernels and the best individual kernel (denoted as BIK) in terms of *macro* F1 score. The number of terms used are 20, 30, and 40, and the number of image sets used are 1000, 1500, and 2000. "SILP", "APP", "SVM1", and "Uniform" denote the performance of kernels combined with the SILP formulation, the approximate formulation, the 1-norm SVM formulation proposed in [12] applied for each label separately, and the case where all kernels are given the same weight, respectively. The subscripts "star" and "clique" denote the way that Laplacian is constructed, and "KCCA" denotes the case where $C = Y^T(YY^T)^{-1}Y$.

| # of labels | 20 | | | 30 | | | 40 | | |
|---|---|---|---|---|---|---|---|---|---|
| # of sets | 1000 | 1500 | 2000 | 1000 | 1500 | 2000 | 1000 | 1500 | 2000 |
| $\text{SILP}_{\text{star}}$ | 0.4396 | 0.4903 | 0.4575 | 0.3852 | 0.4437 | 0.4162 | 0.3768 | 0.4019 | 0.3927 |
| $\text{SILP}_{\text{clique}}$ | 0.4536 | 0.5125 | 0.4926 | 0.4065 | 0.4747 | 0.4563 | 0.4145 | 0.4346 | 0.4283 |
| $\text{SILP}_{\text{KCCA}}$ | 0.3987 | 0.4635 | 0.4477 | 0.3497 | 0.4240 | 0.4063 | 0.3538 | 0.3872 | 0.3759 |
| $\text{APP}_{\text{star}}$ | 0.4404 | 0.4930 | 0.4703 | 0.3896 | 0.4494 | 0.4267 | 0.3900 | 0.4100 | 0.3983 |
| $\text{APP}_{\text{clique}}$ | 0.4510 | 0.5125 | 0.4917 | 0.4060 | 0.4741 | 0.4563 | 0.4180 | 0.4338 | 0.4281 |
| $\text{APP}_{\text{KCCA}}$ | 0.4029 | 0.4805 | 0.4586 | 0.3571 | 0.4313 | 0.4146 | 0.3642 | 0.3914 | 0.3841 |
| SVM1 | 0.3780 | 0.4640 | 0.4356 | 0.3523 | 0.4352 | 0.4200 | 0.3741 | 0.4048 | 0.3955 |
| Uniform | 0.3727 | 0.4703 | 0.4480 | 0.3513 | 0.4410 | 0.4191 | 0.3719 | 0.4111 | 0.3986 |
| BIK | 0.4241 | 0.4515 | 0.4344 | 0.3782 | 0.4312 | 0.3996 | 0.3914 | 0.3954 | 0.3827 |

Table 3: Performance in terms of *micro* F1 score. See the caption of Table 2 for explanations.

| # of labels | 20 | | | 30 | | | 40 | | |
|---|---|---|---|---|---|---|---|---|---|
| # of sets | 1000 | 1500 | 2000 | 1000 | 1500 | 2000 | 1000 | 1500 | 2000 |
| $\text{SILP}_{\text{star}}$ | 0.4861 | 0.5199 | 0.4847 | 0.4472 | 0.4837 | 0.4473 | 0.4277 | 0.4470 | 0.4305 |
| $\text{SILP}_{\text{clique}}$ | 0.5039 | 0.5422 | 0.5247 | 0.4682 | 0.5127 | 0.4894 | 0.4610 | 0.4796 | 0.4660 |
| $\text{SILP}_{\text{KCCA}}$ | 0.4581 | 0.4994 | 0.4887 | 0.4209 | 0.4737 | 0.4532 | 0.4095 | 0.4420 | 0.4271 |
| $\text{APP}_{\text{star}}$ | 0.4852 | 0.5211 | 0.4973 | 0.4484 | 0.4875 | 0.4582 | 0.4355 | 0.4541 | 0.4346 |
| $\text{APP}_{\text{clique}}$ | 0.5013 | 0.5421 | 0.5239 | 0.4673 | 0.5124 | 0.4894 | 0.4633 | 0.4793 | 0.4658 |
| $\text{APP}_{\text{KCCA}}$ | 0.4612 | 0.5174 | 0.5018 | 0.4299 | 0.4828 | 0.4605 | 0.4194 | 0.4488 | 0.4350 |
| SVM1 | 0.4361 | 0.5024 | 0.4844 | 0.4239 | 0.4844 | 0.4632 | 0.3947 | 0.4234 | 0.4188 |
| Uniform | 0.4390 | 0.5096 | 0.4975 | 0.4242 | 0.4939 | 0.4683 | 0.3999 | 0.4358 | 0.4226 |
| BIK | 0.4614 | 0.4735 | 0.4562 | 0.4189 | 0.4484 | 0.4178 | 0.3869 | 0.3905 | 0.3781 |

in combining multiple kernels and exploiting the complementary information contained in different kernels constructed from various features. Moreover, the proposed formulation based on a hypergraph outperforms the classical KCCA consistently.

**SILP versus the Approximate Formulation** In terms of classification performance, we can observe from Tables 2 and 3 that the SILP and the approximate formulations are similar. More precisely, the approximate formulations perform slightly better than SILP in almost all cases. This may be due to the smoothness nature of the formulations and the simplicity of the computational procedure employed in the Nesterov's method so that it is less prone to numerical problems. Figure 1 compares the computation time and the kernel weights of $\text{SILP}_{\text{star}}$ and $\text{APP}_{\text{star}}$. It can be observed that in general the approximate formulation is significantly faster than SILP, especially when the number of labels and the number of image sets are large, while they both yields very similar kernel weights.

## 6    Conclusions and Future Work

We present a multi-label learning formulation that incorporates instance-label correlations by a hypergraph. We formulate the problem in the kernel-induced feature space and propose to learn the kernel matrix in the MKL framework. The resulting formulation leads to a non-smooth min-max problem, and it can be cast as an SILP. We propose an approximate formulation by introducing a smoothing term and show that the resulting formulation is an unconstrained convex problem that can be solved by the Nesterov's method. We demonstrate the effectiveness and efficiency of the method on the task of automated annotation of gene expression pattern images.

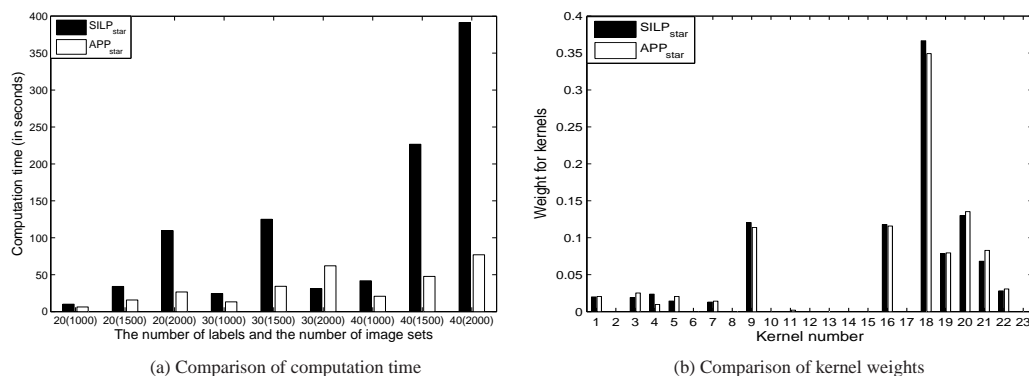

| (a) Comparison of computation time | (b) Comparison of kernel weights |

Figure 1: Comparison of computation time and kernel weights for SILP$_{star}$ and APP$_{star}$. The left panel plots the computation time of two formulations on one partition of the data set as the number of labels and image sets increase gradually, and the right panel plots the weights assigned to each of the 23 kernels by SILP$_{star}$ and APP$_{star}$ on a data set of $40$ labels and $1000$ image sets.

The experiments in this paper focus on the annotation of gene expression pattern images. The proposed formulation can also be applied to the task of multiple object recognition in computer vision. We plan to pursue other applications in the future. Experimental results indicate that the best individual kernel may not lead to a large weight by the proposed MKL formulation. We plan to perform a detailed analysis of the weights in the future.

## Acknowledgements

This work is supported in part by research grants from National Institutes of Health (HG002516 and 1R01-GM079688-01) and National Science Foundation (IIS-0612069 and IIS-0643494).

## References

[1] S. Agarwal, K. Branson, and S. Belongie. Higher order learning with graphs. In *ICML*, pages 17–24, 2006.

[2] D. Zhou, J. Huang, and B. Schölkopf. Learning with hypergraphs: Clustering, classification, and embedding. In *NIPS*, pages 1601–1608. 2007.

[3] Z. H. Zhou and M. L. Zhang. Multi-instance multi-label learning with application to scene classification. In *NIPS*, pages 1609–1616. 2007.

[4] D. R. Hardoon, S. R. Szedmak, and J. R. Shawe-taylor. Canonical correlation analysis: An overview with application to learning methods. *Neural Computation*, 16(12):2639–2664, 2004.

[5] Y. Nesterov. *Introductory Lectures on Convex Optimization: A Basic Course*. Springer, 2003.

[6] Y. Nesterov. Smooth minimization of non-smooth functions. *Mathematical Programming*, 103(1):127–152, 2005.

[7] P. Tomancak and *et al*. Systematic determination of patterns of gene expression during *Drosophila* embryogenesis. *Genome Biology*, 3(12), 2002.

[8] S. Ji, L. Sun, R. Jin, S. Kumar, and J. Ye. Automated annotation of *Drosophila* gene expression patterns using a controlled vocabulary. *Bioinformatics*, 24(17):1881–1888, 2008.

[9] K. Grauman and T. Darrell. Approximate correspondences in high dimensions. In *NIPS*, pages 505–512. 2006.

[10] F. R. K. Chung. *Spectral Graph Theory*. American Mathematical Society, 1997.

[11] S. Schölkopf and A. Smola. *Learning with Kernels: Support Vector Machines,Regularization, Optimization and Beyond*. MIT Press, 2002.

[12] G. R. G. Lanckriet, N. Cristianini, P. Bartlett, L. E. Ghaoui, and M. I. Jordan. Learning the kernel matrix with semidefinite programming. *Journal of Machine Learning Research*, 5:27–72, 2004.

[13] R. Hettich and K. O. Kortanek. Semi-infinite programming: Theory, methods, and applications. *SIAM Review*, 35(3):380–429, 1993.

[14] S. Sonnenburg, G. Rätsch, C. Schäfer, and B. Schölkopf. Large scale multiple kernel learning. *Journal of Machine Learning Research*, 7:1531–1565, July 2006.

[15] K. Mikolajczyk and C. Schmid. A performance evaluation of local descriptors. *IEEE Transactions on Pattern Analysis and Machine Intelligence*, 27(10):1615–1630, 2005.